# Learning Appearance Based Models: Mixtures of Second Moment Experts

**Christoph Bregler and Jitendra Malik**

Computer Science Division
University of California at Berkeley
Berkeley, CA 94720
email: bregler@cs.berkeley.edu, malik@cs.berkeley.edu

## Abstract

This paper describes a new technique for object recognition based on learning appearance models. The image is decomposed into local regions which are described by a new texture representation called "Generalized Second Moments" that are derived from the output of multiscale, multiorientation filter banks. Class-characteristic local texture features and their global composition is learned by a hierarchical mixture of experts architecture (Jordan & Jacobs). The technique is applied to a vehicle database consisting of 5 general car categories (Sedan, Van with back-doors, Van without back-doors, old Sedan, and Volkswagen Bug). This is a difficult problem with considerable in-class variation. The new technique has a 6.5% misclassification rate, compared to eigen-images which give 17.4% misclassification rate, and nearest neighbors which give 15.7% misclassification rate.

## 1 Introduction

Until a few years ago neural network and other statistical learning techniques were not very popular in computer vision domains. Usually such techniques were only applied to artificial visual data or non-mainstream problems such as handwritten digit recognition.

A significant shift has occurred recently with the successful application of appearance-based or viewer-centered techniques for object recognition, supplementing the use of 3D models. Appearance-based schemes rely on collections of images of the object. A principal advantage is that they implicitly capture both shape and photometric information(e.g. surface reflectance variation). They have been most sucessfully applied in the domain of human faces [15, 11, 1, 14] though other 3d objects under fixed lighting have also been considered [13]. View-based representations lend themselves very naturally to learning from examples– principal component analysis[15, 13] and radial basis functions[1] have been used.

Approaches such as principal component analysis (or "eigen-images") use global representations at the image level. The objective of our research was to develop a representation which would

be more 'localist', where representations of different 'parts' of the object would be composed together to form the representation of the object as a whole. This appears to be essential in order to obtain robustness to occlusion and ease of generalization when different objects in a class may have variations in particular parts but not others. A part based view is also more consistent with what is known about human object recognition (Tanaka and collaborators).

In this paper, we propose a domain independent part decomposition using a 2D grid representation of overlapping local image regions. The image features of each local patch are represented using a new texture descriptor that we call "Generalized Second Moments". Related representations have already been successfully applied to other early-vision tasks like stereopsis, motion, and texture discrimination. Class-based local texture features and their global relationships are induced using the "Hierarchical Mixtures of Experts" Architecture (HME) [8].

We apply this technique to the domain of vehicle classification. The vehicles are seen from behind by a camera mounted above a freeway(Figure 1). We urge the reader to examine Figure 3 to see examples of the in class variations in the 5 different categories. Our technique could classify five broader categories with an error of as low as 6.5% misclassification, while the best results using eigen-images and nearest neighbor techniques were 17.4% and 15.7% mis-classification error.

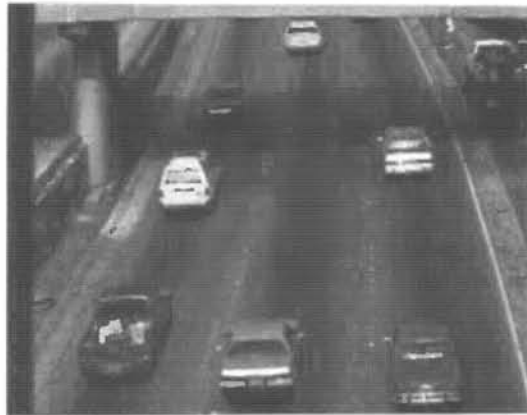

Figure 1: Typical shot of the freeway segment

## 2 Representation

An appearance based representation should be able to capture features that discriminate the different object categories. It should capture both local textural and global structural information. This corresponds roughly to the notion in 3D object models of (i) parts (ii) relationship between parts.

### 2.1 Structural Description

Objects usually can be decomposed into parts. A face consists of eyes, nose, and mouth. Cars are made out of window screens, tail lights, license plates etc. The question is what granularity is appropriate and how much domain knowledge should be exploited. A car could be a single part in a scene, a license plate could be a part, or the letters in the license plate could be the decomposed parts. Eyes, nose, and mouth could be the most important parts of a face for recognition, but maybe other parts are important as well.

It would be advantageous if each part could be described in a decoupled way using a representation that was most appropriate for it. Object classification should be based on these local part descriptions and the relationship between the parts. The partitioning reduces the complexity greatly and invariance to the precise relation between the parts could be achieved.

For our domain of vehicle classification we don't believe it is appropriate to explicitly code any

part decomposition. The kind and number of useful parts might vary across different car makes. The resolution of the images (100x100 pixel) restricts us to a certain degree of granularity. We decided to decompose the image using a 2D grid of overlapping tiles or Gaussian windows but only local classification for each tile region is done. The content of each local tile is represented by a feature vector (next section). The generic grid representation allows the mixture of experts architecture to induce class-based part decomposition, and extract local texture and global shape features. For example the outline of a face could be represented by certain orientation dominances in the local tiles at positions of the face boundary. The eyes are other characteristic features in the tiles.

## 2.2   Local Features

We wanted to extract characteristic features from each local tile. The traditional computer vision approach would be to find edges and junctions. The weakness of these representations is that they do not capture the richness of textured regions, and the hard decision thresholds make the measurement process non–robust.

An alternative view is motivated by our understanding of processing in biological vision systems. We start by convolving image regions with a large number of spatial filters, at various orientations, phases, and scales. The response values of such filters contain much more general information about the local neighborhood, a fact that has now been recognized and exploited in a number of early vision tasks like stereopsis, motion and texture analysis [16, 9, 6, 12, 7].

Although this approach is loosely inspired by the current understanding of processing in the early stages of the primate visual system, the use of spatial filters has many advantages from a pure analytical viewpoint[9, 7]. We use as filter kernels, orientation selective elongated Gaussian derivatives. This enables one to gain the power of orientation specific features, such as edges, without the disadvantage of non-robustness due to hard thresholds. If multiple orientations are present at a single point (e.g. junctions), they are represented in a natural way. Since multiple scales are used for the filters, no *ad hoc* choices have to be made for the scale parameters of the feature detectors. Interestingly the choices of these filter kernels can also be motivated in a learning paradigm as they provide very useful intermediate layer units in convolutional neural networks [3].

The straightforward approach would then be to characterize each image pixel by such a vector of feature responses. However note that there is considerable redundancy in the filter responses– particularly at coarse scales, the responses of filters at neighboring pixels are strongly correlated. We would like to compress the representation in some way. One approach might be to subsample at coarse scales, another might be to choose feature locations with local magnitude maxima or high responses across several directions. However there might be many such interesting points in an image region. It is unclear how to pick the right number of points and how to order them.

Leaving this issue of compressing the filter response representation aside for the moment, let us study other possible representations of low level image data. One way of representing the texture in a local region is to calculate a windowed second moment matrix [5]. Instead of finding maxima of filter responses, the second moments of brightness gradients in the local neighborhood are weighted and averaged with a circular Gaussian window. The gradient is a special case of Gaussian oriented filter banks. The windowed second moment matrix takes into account the response of all filters in this neighborhood. The disadvantage is that gradients are not very orientation selective and a certain scale has to be selected beforehand. Averaging the gradients "washes" out the detailed orientation information in complex texture regions.

Orientation histograms would avoid this effect and have been applied successfully for classification [4]. Elongated families of oriented and scaled kernels could be used to estimate the orientation at each point. But as pointed out already, there might be more than one orientation at each point, and significant information is lost.

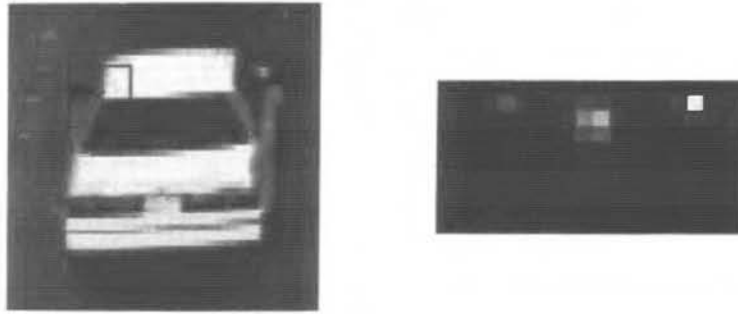

Figure 2: Left image: The black rectangle outlines the selected area of interest. Right image: The reconstructed scale and rotation distribution of the Generalized Second Moments. The horizontal axis are angles between 0 and 180 degrees and the vertical axis are different scales.

## 3   Generalized Second Moments

We propose a new way to represent the texture in a local image patch by combining the filter bank approach with the idea of second moment matrices.

The goal is to compute a feature vector for a local image patch that contains information about the orientation and scale distribution. We compute for each pixel in the image patch the $R$ basis kernel responses (using X-Y separable steerable scalable approximations of a rich filter family). Given a spatial weighting function of the patch (e.g. Gaussian), we compute the covariance matrix of the weighted set of $R$ dimensional vectors. In [2] we show that this covariance matrix can be used to reconstruct for any desired oriented and scaled version of the filter family the weighted sum of all filter response energies:

$$E(\theta, \sigma) = \sum_{x,y} W(x, y)[F_{\theta, \sigma}(x, y)]^2 \tag{1}$$

Using elongated kernels produces orientation/scale peaks, therefore the sum of all orientation/scale responses doesn't "wash" out high peaks. The height of each individual peak corresponds to the intensity in the image. Little noisy orientations have no high energy responses in the sum. $E(\theta, \sigma)$ is somehow a "soft" orientation/scale histogram of the local image patch. Figure 2 shows an example of such a scale/orientation reconstruction based on the covariance matrix (see [2] for details). Three peaks are seen, representing the edge lines along three directions and scales in the local image patch.

This representation greatly reduces the dimensionality without being domain specific or applying any hard decisions. It is shift invariant in the local neighborhood and decouples scale in a nice way. Dividing the $R \times R$ covariance matrix by its trace makes this representation also illumination invariant.

Using a 10x10 grid and a kernel basis of 5 first Gaussian derivatives and 5 second Gaussian derivatives represents each input image as an $10 \cdot 10 \cdot (5 + 1) \cdot 5 = 3000$ dimensional vector (a $5 \times 5$ covariance matrix has $(5 + 1) \cdot 5$ independent parameters). Potentially we could represent the full image with one generalized second moment matrix of dimension 20 if we don't care about capturing the part decomposition.

## 4   Mixtures of Experts

Even if we only deal with the restricted domain of man-made object categories (e.g. cars), the extracted features still have a considerable in-class variation. Different car shapes and poses produce nonlinear class subspaces. Hierarchical Mixtures of Experts (HME by Jordan & Jacobs) are able to model such nonlinear decision surfaces with a soft hierarchical decomposition of the

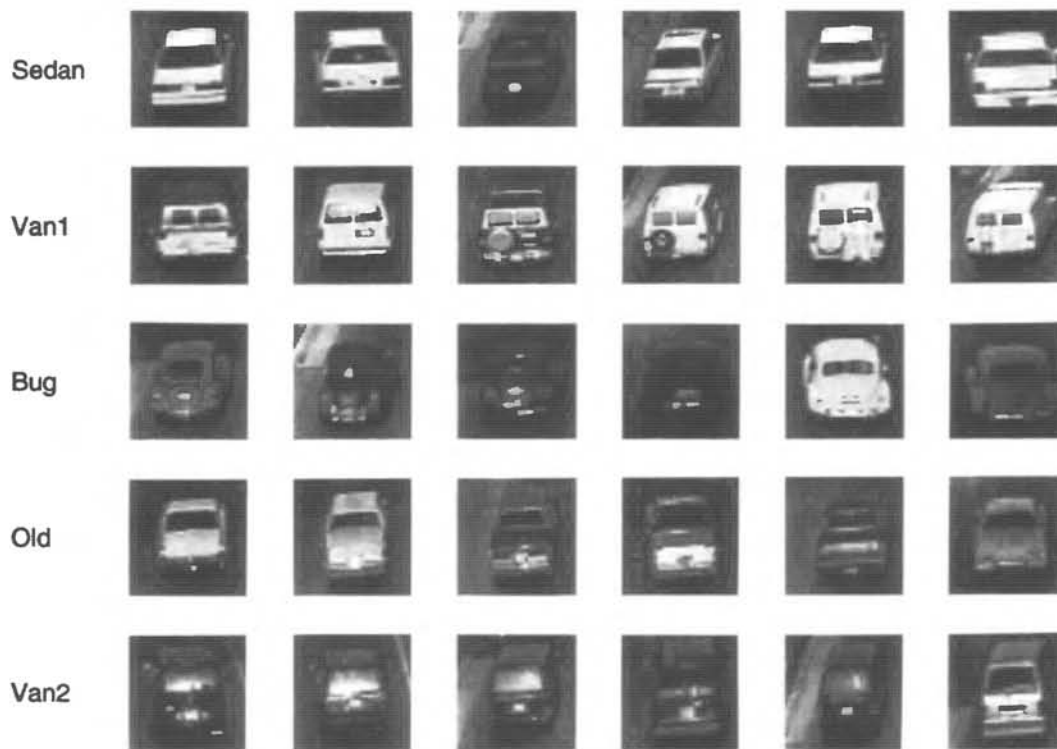

Figure 3: Example images of the five vehicle classes.

feature space and local linear classification experts. Potentially different experts are "responsible" for different object poses or sub-categories.

The gating functions decompose the feature space into a nested set of regions using a hierarchical structure of soft decision boundaries. Each region is the domain for a specific expert that classifies the feature vectors into object categories. We used generalized linear models (GLIM). Given the training data and output labels, the gating functions and expert functions can be estimated using an iterative version of the EM-algorithm. For more detail see [8].

In order to reduce training time and storage requirements, we trained such nonlinear decision surfaces embedded in one global linear subspace. We choose the dimension of this linear subspace to be large enough, so that it captures most of the lower-dimensional nonlinearity (3000 dimensional feature space projected into an 64 dimensional subspace estimated by principal components analysis).

## 5  Experiments

We experimented with a database consisting of images taken from a surveillance camera on a bridge covering normal daylight traffic on a freeway segment (Figure 1). The goal is to classify different types of vehicles. We are able to segment each moving object based on motion cues [10]. We chose the following 5 vehicle classes: Modern Sedan, Old Sedan, Van with back-doors, Van without back-door, and Volkswagen Bug. The images show the rear of the car across a small set of poses (Figure 3). All images are normalized to 100x100 pixel using bilinear interpolation. For this reason the size or aspect ratio can not be used as a feature.

We ran our experiments using two different image representations:

- Generalized Second Moments: A $10 \times 10$ grid was used. Generalized second moments

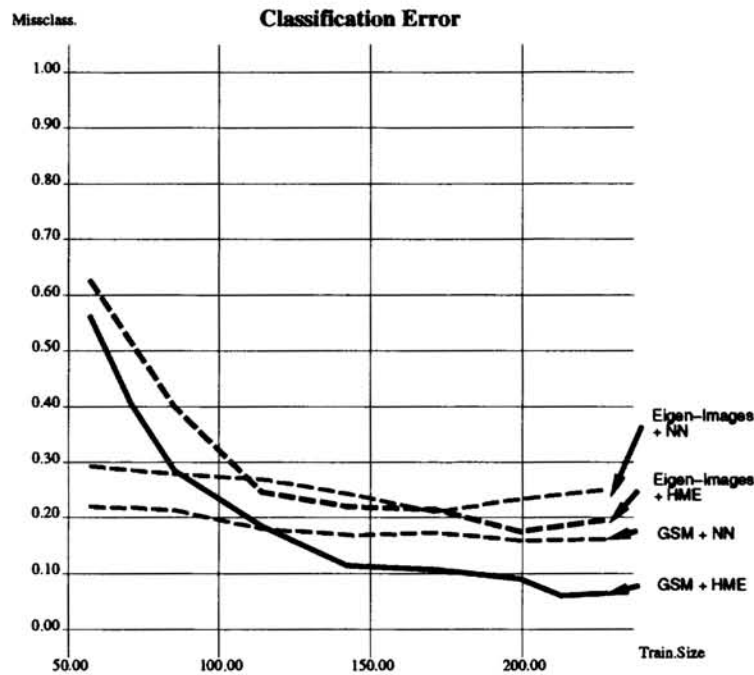

Figure 4: The classification errors of four different techniques. The X-axis shows the size of the training set, and the Y-axis shows the percentage of misclassified test images. HME stands for Hierarchical Mixtures of Experts, GSM stands for Generalized Second Moments, and 1-NN stands for Nearest Neighbors.

were computed [1] using a window of $\sigma = 6$ pixel, and 5 filter bases of 3:1 elongated first and second Gaussian derivatives on a scale range between 0.25 and 1.0.

- Principal Components Analysis ("Eigen-Images"): We used no grid decomposition and projected the global graylevel vector into a 64 dimensional linear space.

Two different classifiers were used:

- HME architecture with 8 local experts.

- A simple 1-Nearest-Neighbor Classifier (1-NN).

Figure 4 shows the classification error rate for all 4 combinations as a function of the size of the training set. Each experiment is run 5 times with different sampling of training and test images [2]. The database consists of 285 example images. Therefore the number of test images are (285−number of training images).

Across all experiments the HME architecture based on Generalized Moments was superior to all other techniques. The best performance with a misclassification of 6.5% was achieved using 228 training images. When fewer than 120 training images are used, the HME architecture performed worse than nearest neighbors.

The most common confusion was between sedans and "old" sedans. The second most confusion was between vans with back-doors, vans without back-doors, and old sedans.

## 6 Conclusion

We have demonstrated a new technique for appearance-based object recognition based on a 2D grid representation, generalized second moments, and hierarchical mixtures of experts. Experiments have shown that this technique has significant better performance than other representation techniques like eigen-images and other classification techniques like nearest neighbors.

We believe that learning such appearance-based representations offers a very attractive methodology. Hand-coding features that could discriminant object categories like the different car types in our database seems to be a nearly impossible task. The only choice in such domains is to estimate discriminating features from a set of example images automatically.

The proposed technique can be applied to other domains as well. We are planning to experiment with face databases, as well as larger car databases and categories to further investigate the utility of hierarchical mixtures of experts and generalized second moments.

**Acknowledgments**   We would like to thank Leo Breiman, Jerry Feldman, Thomas Leung, Stuart Russell, and Jianbo Shi for helpful discussions and Michael Jordan, Lawrence Saul, and Doug Shy for providing code.

## Footnotes

[1] We experimented also with grid sizes between 6x6 to 16x16, and with 8 filter bases and a rectangle window for the second moment statistics without getting significant improvement.

[2] For a given training size $n$ we trained 5 classifiers on 5 different training and test sets and computed the average error rate. The training and test set for each classifier was generated by the same database. The $n$ training examples were randomly sampled from the database, and the remaining examples were used for the test set.

## References

[1] D. Beymer, A. Shashua, and T. Poggio. Example based image analysis and synthesis. *M.I.T. A.I. Memo No. 1431*, Nov 1993.

[2] C. Bregler and J. Malik. Learning Appearance Based Models: Hierarchical Mixtures of Experts Approach based on Generalized Second Moments. Technical Report UCB//CSD-96-897, Comp. Sci. Dep., U.C. Berkeley, http://www.cs/ bregler/soft.html, 1996.

[3] Y. Le Cun, B. Boser, J.S. Denker, S. Solla, R. Howard, and L. Jackel. Back-propagation applied to handwritten zipcode recognition. *Neural Computation*, 1(4), 1990.

[4] W. Freeman and M. Roth. Orientation histograms for hand gesture recognition. In *International Workshop on Automatic Face- and Gesture-Recognition*, 1995.

[5] J. Garding and T. Lindeberg. Direct computation of shape cues using scale-adapted spatial derivative operators. *Int. J. of Computer Vision*, 17, February 1996.

[6] D.J. Heeger. Optical flow using spatiotemporal filters. *Int. J. of Computer Vision*, 1, 1988.

[7] D. Jones and J. Malik. Computational framework for determining stereo correspondence from a set of linear spatial filters. *Image and Vision Computing*, 10(10), 1992.

[8] M.I. Jordan and R. A. Jacobs. Hierarchical mixtures of experts and the em algorithm. *Neural Computation*, 6(2), March 1994.

[9] J.J. Koenderink. Operational significance of receptive field assemblies. *Biol. Cybern.*, 58:163–171, 1988.

[10] D. Koller, J. Weber, and J. Malik. Robust multiple car tracking with occlusion reasoning. In *Proc. Third European Conference on Computer Vision*, pages 189–196, May 1994.

[11] M. Lades, J.C. Vorbrueggen, J. Buhmann, J. Lange, C. von der Malsburg, and R.P. Wuertz. Distortion invariant object recognition in the dynamic link architecure. In *IEEE Transactions on Computers*, volume 42, 1993.

[12] J. Malik and P. Perona. Preattentive texture discrimination with early vision mechanisms. *J. Opt. Soc. Am. A*, 7(5):923–932, 1990.

[13] H. Murase and S.K. Nayar. Visual learning and recognition of 3-d objects from appearance. *Int. J. Computer Vision*, 14(1):5–24, January 1995.

[14] H.A. Rowley, S. Baluja, and T. Kanade. Human face detection in visual scenes. In *NIPS*, volume 8, 1996.

[15] M. Turk and A. Pentland. Eigenfaces for recognition. *Journal of Cognitive Neuroscience*, 3(1):71–86, 1991.

[16] R.A. Young. The gaussian derivative theory of spatial vision: Analysis of cortical cell receptive field line-weighting profiles. Technical Report GMR-4920, General Motors Research, 1985.